# Subject-Independent Magnetoencephalographic Source Localization by a Multilayer Perceptron

**Sung C. Jun**
Biological and Quantum Physics Group
MS-D454, Los Alamos National Laboratory
Los Alamos, NM 87545, USA
*jschan@lanl.gov*

**Barak A. Pearlmutter**
Hamilton Institute
NUI Maynooth
Maynooth, Co. Kildare, Ireland
*barak@cs.may.ie*

## Abstract

We describe a system that localizes a single dipole to reasonable accuracy from noisy magnetoencephalographic (MEG) measurements in real time. At its core is a multilayer perceptron (MLP) trained to map sensor signals and head position to dipole location. Including head position overcomes the previous need to retrain the MLP for each subject and session. The training dataset was generated by mapping randomly chosen dipoles and head positions through an analytic model and adding noise from real MEG recordings. After training, a localization took 0.7 ms with an average error of 0.90 cm. A few iterations of a Levenberg-Marquardt routine using the MLP's output as its initial guess took 15 ms and improved the accuracy to 0.53 cm, only slightly above the statistical limits on accuracy imposed by the noise. We applied these methods to localize single dipole sources from MEG components isolated by blind source separation and compared the estimated locations to those generated by standard manually-assisted commercial software.

## 1 Introduction

The goal of MEG/EEG localization is to identify and measure the signals emitted by electrically active brain regions. A number of methods are in widespread use, most assuming dipolar sources (Hämäläinen et al., 1993). Recently MLPs (Rumelhart et al., 1986) have become popular for building fast dipole localizers (Abeyratne et al., 1991; Kinouchi et al., 1996). Since it is easy to use a forward model to create synthetic data consisting of dipole locations and corresponding sensor signals, one can train a MLP on the inverse problem. Hoey et al. (2000) took EEG measurements for both spherical and realistic head models and trained MLPs on randomly generated noise-free datasets. Integrated approaches to the EEG/MEG dipole source localization, in which the trained MLPs are used as initializers for iterative methods, have also been studied (Jun et al., 2002) along with distributed output representations (Jun et al., 2003). Interestingly, all work to date trained with a fixed head model. However, for MEG, head movement relative to the fixed sensor array is very difficult to avoid, and even with heroic measures (bite bars) the position of the head relative to the sensor array varies from subject to subject and session to session. This either results in significant localization error (Kwon et al., 2002), or requires laborious retraining and

revalidation of the system.

We propose an augmented system which takes head position into account, yet remains able to localize a single dipole to reasonable accuracy within a fraction of a millisecond on a standard PC, even when the signals are contaminated by considerable noise. The system uses a MLP trained on random dipoles and random head positions, which takes as inputs both the coordinates of the center of a sphere fitted to the head and the sensor measurements, uses two hidden layers, and generates the source location (in Cartesian coordinates) as its output. Adding head position as an extra input overcomes the primary practical limitation of previous MLP-based MEG localization systems: the need to retrain the network for each new head position.

We use an analytical model of quasi-static electromagnetic propagation through a spherical head to map randomly chosen dipoles and head positions to superconducting quantum interference device (SQUID) sensor activities according to the sensor geometry of a 4D Neuroimaging Neuromag-122 MEG system, and trained a MLP to invert this mapping in the presence of real brain noise. To improve the localization accuracy we use a hybrid MLP-start-LM method, in which the MLP's output provides the starting point for a Levenberg-Marquardt (LM) optimization (Press et al., 1988). We use the MLP and MLP-start-LM methods to localize single-dipole sources from actual MEG signal components isolated by a blind source separation (BSS) algorithm (Vigário et al., 2000; Tang et al., 2002) and compare the results with the output of standard interactive commercial localization software.

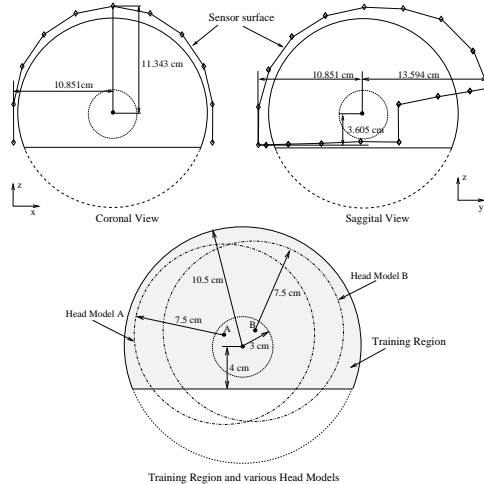

Figure 1: Sensor surface and training region. The center of the spherical head model was varied within the given region. Diamonds denote sensors.

Section 2 describes our synthetic data, the forward model, the noise used to additively contaminate the training data, and the MLP structure. Section 3 presents the localization performance of both the MLP and MLP-start-LM, and compares them with various conventional LM methods. In Section 3.2, comparative localization results for our proposed methods and standard Neuromag commercial software on actual BSS-separated MEG signals are presented.

## 2   Data and MLP structure

We constructed noisy data using the procedure of Jun et al. (2002), except that an additional input was associated with each exemplar, namely the $(x, y, z)$ coordinates of the center of a sphere fitted to the head, and the forward model was modified to account for this offset. Each exemplar thus consisted of the $(x, y, z)$ coordinates of the center of a sphere fitted to the head, sensor activations generated by a forward model, and the target dipole location.[1]

We made two datasets: one for training and another for testing. Centers of spherical head

models in the training set were drawn from a ball of radius 3 cm centered 4 cm above the bottom of the training region,[2] as shown in Figure 1. The dipoles in the training set were drawn uniformly from a spherical region centered at the corresponding center, with a radius of 7.5 cm, and truncated at the bottom. Their moments were drawn uniformly from vectors of strength $\leq$200 nAm. The corresponding sensor activations were calculated by adding the results of a forward model and a noise model. To check the performance of the network during training, a test set was generated in the same fashion as the training set. We used the sensor geometry of a 4D Neuroimaging Neuromag-122 whole-head gradiometer (Ahonen et al., 1993) and a standard analytic model of quasistatic electromagnetic propagation in a spherical head (Jun et al., 2002).

This work could be easily extended to a more realistic head model. In that case the integral equations are solved by the boundary element method (BEM) or the finite element method (FEM) numerically (Hämäläinen et al., 1993). The human skull phantom study in Leahy et al. (1998) shows that the fitted spherical head model for MEG localization is slightly inferior in accuracy to the realistic head model numerically calculated by BEM. In forward calculation, a spherical head model has some advantages: it is more easily implemented and is much faster. Despite its inferiority in terms of localization accuracy, we use a spherical head model in this work.

In order to properly compare the performance of various localizers, we need a dataset for which we know the ground truth, but which contains the sorts of noise encountered in actual MEG recordings. To this end, we measured real brain noise and used it to additively contaminate synthetic sensor readings (Jun et al., 2002). This noise was taken, unaveraged, from MEG recordings during periods in which the brain region of interest in the experiment was quiescent, and therefore included all sources of noise present in actual data: brain noise, external noise, sensor noise, etc. This had a RMS (square root of mean square) magnitude of roughly $P^n = 50\text{--}200\,\text{fT/cm}$, where we measure the SNR of a dataset using the ratios of the powers in the signal and noise, SNR (in dB) $= 20\log_{10} P^s/P^n$, where $P^s$ and $P^n$ are the RMS sensor readings from the dipole and noise, respectively. The datasets used for training and testing were made by adding the noise to synthetic sensor activations generated by the forward model, and exemplars whose resulting SNR was below $-4$ dB were rejected.

The MLP charged with approximating the inverse mapping had an input layer of 125 units consisting of the three Cartesian coordinate of the center of the sphere fitted to the head, and the 122 sensor activations. It had two hidden layers with 320 and 30 units respectively, and an output layer of three units representing the Cartesian coordinates of the fitted dipole. The output units had linear activation functions, while the hidden unit had hyperbolic tangent activation functions. Adjacent layers were fully connected, with no cut-through connections. The 122 sensor activation inputs were scaled to an RMS value of 0.5, and the target outputs were scaled into $[-1, +1]$. The network weights were initialized with uniformly distributed random values between $\pm 0.1$, and online stochastic gradient decent with no momentum and an empirically chosen constant of proportionality was used for optimization.

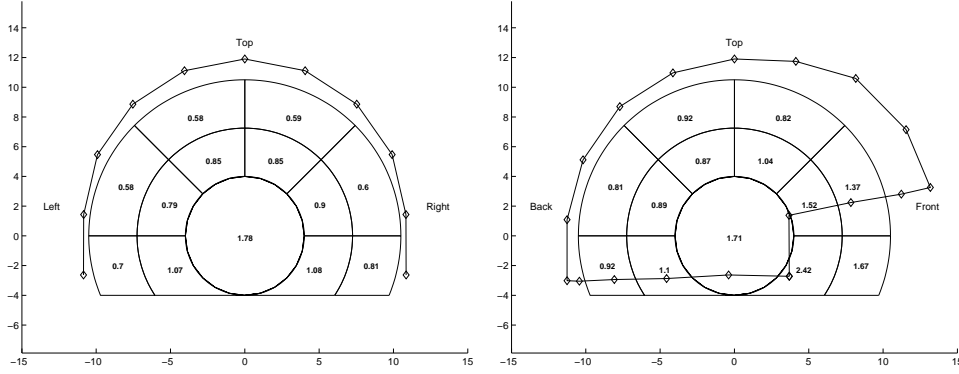

Figure 2: Mean localization errors of the trained MLP as a function of correct dipole location, binned into regions. All units are in cm. Left: Coronal cross section. Right: Sagittal cross section.

# 3 Results and discussion

## 3.1 Training and localization results

Datasets of 100,000 (training) and 25,000 (testing) patterns, all contaminated by real brain noise, were constructed. As is typical, the incremental gains per epoch decrease exponentially with training. From the training curves (not shown) it is evident that additional training would have further decreased the error, but we nonetheless stopped after 1000 epochs, which took about three days on 2.8 GHz Intel Xeon CPU.

We investigated localization error distributions over various regions of interest. We considered two cross sections (coronal and sagittal views) with width of 2 cm, and each of these was divided into 19 regions,

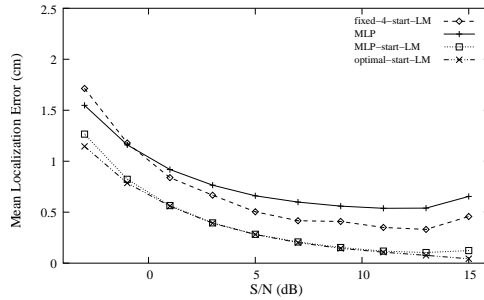

Figure 3: Mean localization error *vs.* SNR. MLP, MLP-start-LM, and optimal-start-LM were tested on signals from 25,000 random dipoles, contaminated by real brain noise.

as shown in Figure 2. We extracted the noisy signals and the corresponding dipoles from testing datasets. For each region 49–500 patterns were collected. A dipole localization was performed using the trained MLP, and the average localization error for each region was calculated. Figure 2 shows the localization error distribution over two cross sections. In general, dipoles closer to the sensor surface were better localized.

We compared various automatic localization methods, most of which consist of LM used in different ways:

- MLP-start-LM
  LM was started with the trained MLP's output.

- fixed-4-start-LM
  LM was tuned for good performance using restarts at the four fixed initial points $(0, 0, 6)$, $(-5, 2, -1)$, $(5, 2, -1)$, and $(0, -5, -1)$, in units of cm relative to the center of the spherical head model. The best result among four results was chosen.

Table 1: Comparison of performance on real brain noise test set of Levenberg-Marquardt source localizers with three LM restarts strategies, the trained MLP, and a hybrid system. Each number is an average over 25,000 localizations, so the error bars are negligible.

| Algorithm | Computation time (ms) | Localization error (cm) |
|---|---|---|
| fixed-4-start-LM | 120 | 0.83 |
| random-20-start-LM | 663 | 0.54 |
| optimal-start-LM | 14 | 0.49 |
| MLP | 0.7 | 0.90 |
| MLP-start-LM | 15 | 0.53 |

- random-$n$-start-LM
  LM was restarted with $n$ random (uniformly distributed) points within the spherical head model. We checked how many restarts were needed to match the accuracy of the MLP-start-LM, yielding $n = 20$, which is the same as in Jun et al. (2002).

- optimal-start-LM
  LM was started with the known exact dipole source location.

Figure 3 shows the localization performance as a function of SNR for fixed-4-start-LM, optimal-start-LM, the trained MLP, and MLP-start-LM. Optimal-start-LM shows the best localization performance across the whole range of SNRs, but the hybrid system shows almost the same performance as optimal-start-LM except at very high SNRs, while the trained MLP is more robust to noise than fixed-4-start-LM. In this experiment, most of the sources with very high SNR were superficial, located around the upper neck or back of the head. These sorts of sources are often very hard to localize well, as it is easy to become trapped in a local minimum (Jun et al., 2002). It is expected that, under these conditions, a better initial guess than the MLP output (which are 0.7 cm on average from the exact source) would be required to obtain near-optimal performance from LM.

A grand summary, averaged across various SNR conditions, is shown in Table 1. The trained MLP is fastest, and its hybrid system is about 40× faster than random-20-start-LM, while the hybrid system is about 9× faster, yet more accurate than, fixed-4-start-LM. This means that MLP-start-LM was about two times faster than might be naively expected.

## 3.2  Localization on real MEG signals and comparison with commercial software

The sensors in MEG systems have poor signal-to-noise ratios (SNRs) for single-trial data, since MEG data is strongly contaminated by various noises. Blind source separation of MEG data segregates noise from signal (Vigário et al., 2000; Tang et al., 2000a; Sander et al., 2002), raising the SNR sufficiently to allow single-trial analysis (Tang et al., 2000b). Even though the sensor attenuation vectors of the BSS-separated components can be well localized to equivalent current dipoles (Vigário et al., 2000; Tang et al., 2002), the recovered field maps can be quite noisy. We applied the MLP and MLP-start-LM to localize single dipolar sources from various actual BSS-separated MEG signals.[3] The xfit program

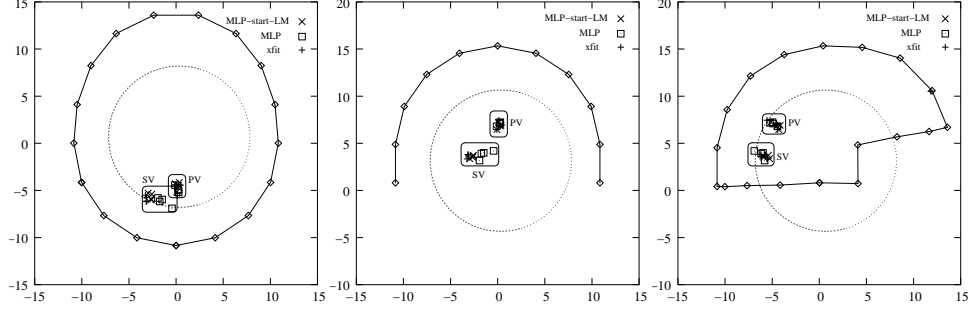

Figure 4: Dipole source localization results of Neuromag software (xfit), our MLP, MLP-start-LM for four BSS-separated primary visual and four secondary visual MEG signal components of S01, over four sorts of tasks. PV and SVdenote primary visual source and secondary visual source, respectively. Left: Axial view. Center: Coronal view. Right: Sagittal view. The outer surface denotes the sensor surface, and diamonds on this surface denote sensors. The inner surface denotes a spherical head model fit to the subject.

(standard commercial software bundled with the 4D Neuroimaging Neuromag-122 MEG system) is compared with the methods developed here.

A field map of each component was scaled to an RMS of 0.5 and inputed to the trained MLP. Their MLP's outputs were scaled back to their dipole location vectors and were used for initializing LM. Figure 4 shows the dipole locations estimated by the MLP, MLP-start-LM, and Neuromag's xfit software, for two sorts of sensory sources: primary visual sources and secondary visual sources, respectively, over four tasks in subject S01. In Figure 5, the estimated dipole locations are shown for somatosensory sources over three different subjects. Each figure consists of three viewpoints: axial (x-y plane), coronal (x-z plane), and sagittal (y-z plane). The center of a fitted spherical head model (S01: trump card task) is $(0.335, 0.698, 3.157)$. All units are in cm. All dipole locations estimated by the MLP and MLP-start-LM are clustered within about 3 cm, and about 0.7 cm, of xfit's results, respectively. We see that the primary visual sources are more consistently localized, across all four tasks, than the secondary visual sources. The secondary sources also had more variable stimulus-locked average time courses (Tang and Pearlmutter, 2003). It is noticeable that somatosensory sources on the right hemisphere are localized poorly by the MLP, but well localized by the hybrid method. Even though the auditory sources are the weakest (not shown here), *i.e.* have the lowest SNRs, they are reasonably well localized.

While the MLP-estimated location is about 1.16 cm ($|dx| \approx 0.90$, $|dy| \approx 0.57$, $|dz| \approx 0.46$) on average ($N = 14$) from those of xfit, the hybrid method's result is about 0.35 cm ($|dx| \approx 0.20$, $|dy| \approx 0.22$, $|dz| \approx 0.10$) from xfit's estimated location. Considering that xfit had extra information, namely the identity of a subset of the sensors to use, this hybrid method result is believed to be almost as good as the xfit result. The trained MLP and the hybrid method are applicable to actual MEG signals, and seem to offer comparable and perhaps superior localization relative to xfit, with clear advantages in both speed and in the lack of required human interaction or subjective human input.

---

SOBI was performed on continuous 122-channel data collected during the entire period of the experiment. It generated 122 components, each a one-dimensional time series with an associated field map. Event triggered averages were calculated from their continuous single-trial time series for all 122 separated components. A dipole fitting method was applied to the identified neural components. The input to the dipole fitting algorithm of xfit was the field map and the output was the location of ECDs. From all separated components for four subjects and four sorts of tasks taken as in Tang et al. (2002). only fourteen components were localized and compared. For further experimental details and a detailed SOBI algorithm, see Tang et al. (2002).

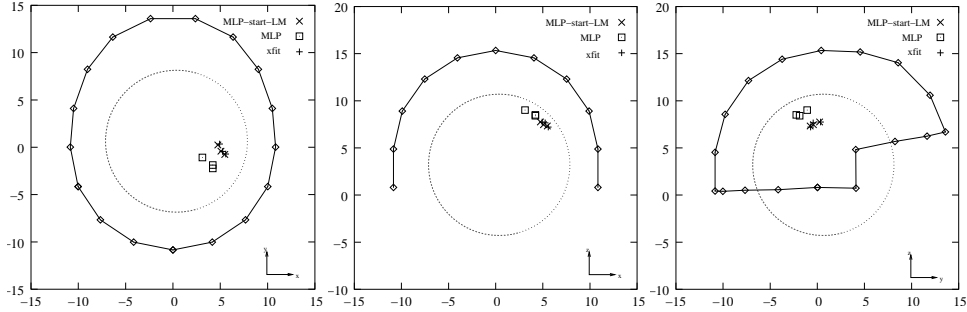

Figure 5: Dipole source localization results of Neuromag software (xfit), our MLP, MLP-start-LM for three real BSS-separated somatosensory MEG signal components from the transverse patterning task over three different subjects (S01, S02, S03). Even the center of a fitted spherical head model is varied over three subjects, the only fitted sphere of subject S01 transverse patterning task, centered at $(0.373, 0.642, 3.205)$, is depicted. Left: Axial view. Center: Coronal view. Right: Sagittal view. The outer surface denotes the sensor surface, and diamonds on this surface denote sensors. The inner surface denotes a spherical head model fit to the subject.

## 4  Conclusion

We propose the inclusion of a head position input for MLP-based MEG dipole localizers. This overcomes the limitation of previous MLP-based MEG localization systems, namely the need to retrain the network for each session or subject. Experiments showed that the trained MLP was far faster, albeit slightly less accurate, than fixed-4-start-LM. This motivated us to construct a hybrid system, MLP-start-LM, which improves the localization accuracy while reducing the computational burden to less than one ninth than that of fixed-4-start-LM. This hybrid method was comparable in accuracy to random-20-start-LM, at 1/40-th the computation burden, which is about two times faster than might be naively expected. Over the whole range of SNRs, the hybrid system showed almost as good performance in accuracy and computation time as the hypothetical optimal-start-LM.

We applied the MLP and MLP-start-LM to localize single dipolar sources from actual BSS-separated MEG signals, and compared these with the results of the commercial Neuromag program xfit. The MLP yielded dipole locations close to those of xfit, and MLP-start-LM gave locations that were even closer to those of xfit.

In conclusion, our MLP can itself serve as a reasonably accurate real-time MEG dipole localizer, even when the head position changes regularly. This MLP also constitutes an excellent dipole guessor for LM. Because this MLP receives a head position input, the need to retrain for various subjects or sessions has been eliminated without sacrificing the many advantages of the universal approximator direct inverse approach to localization.

**Acknowledgements**

This work was supported by NSF CAREER award 97-02-311, the Mental Illness and Neuroscience Discovery Institute, a gift from the NEC Research Institute, NIH grant 2 R01 EB000310-05, and Science Foundation Ireland grant 00/PI.1/C067. We would like to thank Guido Nolte for help with the forward model, Michael Weisend for allowing us to use his data, and Michael Weisend, Akaysha Tang, and Natalie Malaszenko for providing experimental details.

## Footnotes

[1]Given the sensor activations and a dipole location, the minimum error dipole moment can be calculated analytically (Hämäläinen et al., 1993). Therefore, although the dipoles used in generating the dataset had both location and moment, the moments were not included in the datasets used for training or testing.

[2]Fitted spheres from twelve subjects performing various tasks on a 4D Neuroimaging Neuromag-122 MEG system were collected, and this distribution of head positions was chosen to include all twelve cases. Just as the position of the center of the head varies from session to session and subject to subject, so does head orientation and radius. Because a sphere is rotationally symmetric, our forward model is insensitive to orientation, and similarly the external magnetic field caused by a dipole in a homogeneous sphere is invariant to the sphere's radius. On the other hand, the noise process would not be invariant to orientation or radius, so we might expect a slight increase in performance if the network had orientation and radius available as inputs, rather than just the position of the center.

[3]Continuous 300 Hz MEG data for four right-handed subjects was collected using a cognitive protocol developed by Michael P. Weisend, band-pass filtered at 0.03–100 Hz, separated using second order blind identification algorithm (SOBI), and scanned for neuronal sources of interest. The following four visual reaction time tasks were performed by each subject: stimulus pre-exposure task, trump card task, elemental discrimination task, and transverse patterning task. For each subject, all four experiments were performed on the same day, but each in a separate session. Subjects were permitted to move their heads between experiments.

# References

Abeyratne, U. R., Kinouchi, Y., Oki, H., Okada, J., Shichijo, F., and Matsumoto, K. (1991). Artificial neural networks for source localization in the human brain. *Brain Topography*, 4:3–21.

Ahonen, A. I., Hämäläinen, M. S., Knuutila, J. E. T., Kajola, M. J., Laine, P. P., Lounasmaa, O. V., Parkkonen, L. T., Simola, J. T., and Tesche, C. D. (1993). 122-channel SQUID instrument for investigating the magnetic signals from the human brain. *Physica Scripta*, T49:198–205.

Hämäläinen, M., Hari, R., Ilmoniemi, R. J., Knuutila, J., and Lounasmaa, O. V. (1993). Magnetoencephalography—theory, instrumentation, and applications to noninvasive studies of the working human brain. *Rev. Modern Physics*, 65:413–497.

Hoey, G. V., Clercq, J. D., Vanrumste, B., de Walle, R. V., Lemahieu, I., D'Havé, M., and Boon, P. (2000). EEG dipole source localization using artificial neural networks. *Phys. Med. Biol.*, 45:997–1011.

Jun, S. C., Pearlmutter, B. A., and Nolte, G. (2002). Fast accurate MEG source localization using a multilayer perceptron trained with real brain noise. *Physics in Medicine and Biology*, 47(14):2547–2560.

Jun, S. C., Pearlmutter, B. A., and Nolte, G. (2003). MEG source localization using a MLP with a distributed output representation. *IEEE Transactions on Biomedical Engineering*, 50(6):786–789.

Kinouchi, Y., Ohara, G., Nagashino, H., Soga, T., Shichijo, F., and Matsumoto, K. (1996). Dipole source localization of MEG by BP neural networks. *Brain Topography*, 8:317–321.

Kwon, H., Lee, Y. H., Kim, J. M., Park, Y. K., and Kuriki, S. (2002). Localization accuracy of single current dipoles from tangential components of auditory evoked fields. *Phys. Med. Biol.*, 47:4145–4154.

Leahy, R. M., Mosher, J. C., Spencer, M. E., Huang, M. X., and Lewine, J. D. (1998). A study of dipole localization accuracy for MEG and EEG using a human skull phantom. *Electroencephalography and clinical neurophysiology*, 107(2):159–173.

Press, W. H., Flannery, B. P., Teukolsky, S. A., and Verrerling, W. T. (1988). *Numerical Recipes in C*. Cambridge University Press.

Rumelhart, D. E., Hinton, G. E., and Williams, R. J. (1986). Learning representations by back–propagating errors. *Nature*, 323:533–536.

Sander, T. H., Wübbeler, G., Lueschow, A., Curio, G., and Trahms, L. (2002). Cardiac artifact subspace identification and elimination in cognitive MEG data using time-delayed decorrelation. *IEEE Transactions on Biomedical Engineering*, 49:345–354.

Tang, A. C. and Pearlmutter, B. A. (2003). Independent components of magnetoencephalography: Localization and single-trial response onset detection. In Lu, Z.-L. and Kaufman, L., editors, *Magnetic Source Imaging of the Human Brain*, pages 159–201. Lawrence Erlbaum Associates.

Tang, A. C., Pearlmutter, B. A., Malaszenko, N. A., Phung, D. B., and Reeb, B. C. (2002). Independent components of magnetoencephalography: Localization. *Neural Computation*, 14(8):1827–1858.

Tang, A. C., Pearlmutter, B. A., Zibulevsky, M., and Carter, S. A. (2000a). Blind separation of multichannel neuromagnetic responses. *Neurocomputing*, 32–33:1115–1120.

Tang, A. C., Pearlmutter, B. A., Zibulevsky, M., Hely, T. A., and Weisend, M. P. (2000b). An MEG study of response latency and variability in the human visual system during a visual-motor integration task. In *Advances in Neural Information Processing Systems 12*, pages 185–191. MIT Press.

Vigário, R., Särelä, J., Jousmäki, V., Hämäläinen, M., and Oja, E. (2000). Independent component approach to the analysis of EEG and MEG recordings. *IEEE Transactions on Biomedical Engineering*, 47(5):589–593.
